# Fast and Balanced: Efficient Label Tree Learning for Large Scale Object Recognition

**Jia Deng**[1,2]**, Sanjeev Satheesh**[1]**, Alexander C. Berg**[3]**, Li Fei-Fei**[1]
Computer Science Department, Stanford University[1]
Computer Science Department, Princeton University[2]
Computer Science Department, Stony Brook University[3]

## Abstract

We present a novel approach to efficiently learn a label tree for large scale classification with many classes. The key contribution of the approach is a technique to simultaneously determine the structure of the tree and learn the classifiers for each node in the tree. This approach also allows fine grained control over the efficiency vs accuracy trade-off in designing a label tree, leading to more balanced trees. Experiments are performed on large scale image classification with 10184 classes and 9 million images. We demonstrate significant improvements in test accuracy and efficiency with less training time and more balanced trees compared to the previous state of the art by Bengio et al.

## 1   Introduction

Classification problems with many classes arise in many important domains and pose significant computational challenges. One prominent example is recognizing tens of thousands of visual object categories, one of the grand challenges of computer vision. The large number of classes renders the standard one-versus-all multiclass approach too costly, as the complexity grows linearly with the number of classes, for both training and testing, making it prohibitive for practical applications that require low latency or high throughput, *e.g.* those in robotics or in image retrieval.

Classification with many classes has received increasing attention recently and most approaches appear to have converged to tree based models [2, 3, 9, 1]. In particular, Bengio *et al.* [1] proposes a *label tree* model, which has been shown to achieve state of the art performance in testing. In a label tree, each node is associated with a subset of class labels and a linear classifier that determines which branch to follow. In performing the classification task, a test example travels from the root of the tree to a leaf node associated with a single class label. Therefore for a well balanced tree, the time required for evaluation is reduced from $O(DK)$ to $O(D \log K)$, where $K$ is the number of classes and $D$ is the feature dimensionality. The technique can be combined with an embedding technique, so that the evaluation cost can be further reduced to $O(\tilde{D} \log K + D\tilde{D})$ where $\tilde{D} \ll D$ is an embedded label space.

Despite the success of label trees in addressing testing efficiency, the learning technique, critical to ensuring good testing accuracy and efficiency, has several limitations. Learning the tree structure (determining how to split the classes into subsets) involves first training one-vs-all classifiers for all $K$ classes to obtain a confusion matrix, and then using spectral clustering to split the classes into disjoint subsets. First, learning one-vs-all classifiers is costly for large number of classes. Second, the partitioning of classes does not allow overlap, which can be unnecessarily difficult for classification. Third, the tree structure may be unbalanced, which can result in sub-optimal test efficiency.

In this paper, we address these issues by observing that (1)determining the partition of classes and learning a classifier for each child can be performed jointly, and (2)allowing overlapping of class

labels among children leads to an efficient optimization that also enables precise control of the accuracy vs efficiency trade-off, which can in turn guarantee balanced trees. This leads to a novel label tree learning technique that is more efficient and effective. Specifically, we eliminate the one-vs-all training step while improving both efficiency *and* accuracy in testing.

## 2    Related Work

Our approach is directly motivated by the label tree embedding technique proposed by Bengio *et al*. in [1], which is among the few approaches that address sublinear testing cost for multi-class classification problems with a large number of classes and has been shown to outperform alternative approaches including Filter Tree [2] and Conditional Probability Tree(CPT) [3]. Our contribution is a new technique to achieve more efficient and effective learning for label trees. For a comprehensive discussion on multi-class classification techniques, we refer the reader to [1].

Classifying a large number of object classes has received increasing attention in computer vision as datasets with many classes such as ImageNet [7] become available. One line of work is concerned with developing effective feature representations [13, 16, 15, 10] and achieving state of the art performances. Another direction of work, explores methods for exploiting the structure *between* object classes. In particular, it has been observed that object classes can be organized in a tree-like structure both semantically and visually [9, 11, 6], making tree based approaches especially attractive. Our work follows this direction, focusing on effective learning methods for building tree models.

Our framework of explicitly controlling accuracy or efficiency is connected to Weiss *et al*.'s work [14] on building a cascade of graphical models with increasing complexity for structured prediction. Our work differs in that we reduce the *label* space instead of the model space.

## 3    Label Tree and Label Tree Learning by Bengio *et al*.

Here we briefly review the label tree learning technique proposed by Bengio *et al*. and then discuss the limitations we attempt to address.

A label tree is a tree $T = (V, E)$ with nodes $V$ and edges $E$. Each node $r \in V$ is associated with a set of class labels $\kappa(r) \subseteq \{1, \ldots, K\}$ . Let $\sigma(r) \subset V$ be the its set of children. For each child $c$, there is a linear classifier $w_c \in \mathbb{R}^D$ and we require that its label set is a subset of its parent's, that is, $\kappa(c) \subseteq \kappa(r), \forall c \in \sigma(r)$.

To make a prediction given an input $x \in \mathbb{R}^D$, we use Algorithm 1. We travel from the root until we reach a leaf node, at each node following the child that has the largest classifier score. There is a slight difference than the algorithm in [1] in that the leaf node is not required to have only one class label. If there is more than one label, an arbitrary label from the set is predicted.

---

**Algorithm 1** Predict the class of $x$ given the root node $r$

---
$s \leftarrow r$.
**while** $\sigma(s) \neq \emptyset$ **do**
    $s \leftarrow \arg\max_{c \in \sigma(s)} w_c^T x$
**end while**
**return**  an arbitrary $k \in \kappa(s)$ or *NULL* if $\kappa(s) = \emptyset$.

---

Learning the tree structure is a fundamentally hard problem because brute force search for the optimal combination of tree structure and classifier weights is intractable. Bengio *et al*. [1] instead propose to solve two subproblems: learning the tree structure and learning the classifier weights. To learn the tree structure, $K$ one versus all classifiers are trained first to obtain a confusion matrix $C \in \mathbb{R}^{K \times K}$ on a validation set. The class labels are then clustered into disjoint sets by spectral clustering with the confusion between classes as affinity measure. This procedure is applied recursively to build a complete tree. Given the tree structure, all classifier weights are then learned jointly to optimize the misclassification loss of the tree.

We first analyze the cost of learning by showing that training, with m examples, K classes and D dimensional feature, costs O(mDK). Assume optimistically that the optimization algorithm converges

after only one pass of the data and that we use first order methods that cost $O(D)$ at each iteration, with feature dimensionality $D$. Therefore learning one versus all classifiers costs $O(mDK)$. Spectral clustering only depends on $K$ and does not depend on $D$ or $m$, and therefore its cost is negligible. In learning the classifier weights on the tree, each training example is affected by only the classifiers on its path, $i.e.$ $O(Q\log K)$ classifiers, where $Q \ll K$ is the number of children for each node. Hence the training cost is $O(mDQ\log K)$. This analysis indicates that learning $K$ one versus all classifiers dominates the cost. This is undesirable in large scale learning because with bounded time, accommodating a large number of classes entails using less expressive and lower dimensional features.

Moreover, spectral clustering only produces disjoint subsets. It can be difficult to learn a classifier for disjoint subsets when examples of certain classes cannot be reliably classified to one subset. If such mistakes are made at higher level of the tree, then it is impossible to recover later. Allowing overlap potentially yields more flexibility and avoids such errors. In addition, spectral clustering does not guarantee balanced clusters and thus cannot ensure a desired speedup. We seek a novel learning technique that overcomes these limitations.

# 4 New Label Tree Learning

To address the limitations, we start by considering simple and less expensive alternatives of generating the splits. For example, we can sub-sample the examples for one-vs-all training, or generate the splits randomly, or use a human constructed semantic hierarchy($e.g.$ WordNet [8]). However, as shown in [1], improperly partitioning the classes can greatly reduce testing accuracy and efficiency. To preserve accuracy, it is important to split the classes such that they can be easily separated. To gain efficiency, it is important to have balanced splits.

We therefore propose a new technique that jointly learns the splits and classifier weights. By tightly coupling the two, this approach eliminates the need of one-vs-all training and brings the total learning cost down to $O(mDQ\log K)$. By allowing overlapping splits and explicitly modeling the accuracy and efficiency trade-off, this approach also improves testing accuracy and efficiency.

Our approach processes one node of the tree a time, starting with the root node. It partitions the classes into a fixed number of child nodes and learns the classifier weights for each of the children. It then recursively repeats for each child.

In learning a tree model, accuracy and efficiency are inherently conflicting goals and some trade-off must be made. Therefore we pose the optimization problem as maximizing efficiency given a constraint on accuracy, $i.e.$ requiring that the error rate cannot exceed a certain threshold. Alternatively one can also optimize accuracy given efficiency constraints. We will first describe the accuracy constrained optimization and then briefly discuss the efficiency constrained variant. In practice, one can choose between the two formulations depending on convenience.

For the rest of this section, we first express all the desiderata in one single optimization problem(Sec. 4.1), including defining the optimization variables(classifier weights and partitions), objectives(efficiency) and constraints(accuracy). Then in Sec. 4.2& 4.3 we show how to solve the main optimization by alternating between learning the classifier weights and determining the partitions. We then summarize the complete algorithm(Sec. 4.4) and conclude with an alternative formulation using efficiency constraints(Sec. 4.5).

## 4.1 Main optimization

Formally, let the current node $r$ represent classes labels $\kappa(r) = \{1, \ldots, K\}$ and let $Q$ be the specified number of children we wish to follow. The goal is to determine: (1)a partition matrix $P \in \{0,1\}^{Q \times K}$ that represents the assignment of classes to the children, $i.e.$ $P_{qk} = 1$ if class label $k$ appear in child $q$ and $P_{qk} = 0$ otherwise; (2)the classifier weights $w \in \mathbb{R}^{D \times Q}$, where a column $w_q$ is the classifier weights for child $q \in \sigma(r)$,

We measure accuracy by examining whether an example is classified to the correct child, $i.e.$ a child that includes its true class label. Let $x \in \mathbb{R}^D$ be a training example and $y \in \{1, \ldots, K\}$ be its true label. Let $\hat{q} = \arg\max_{q \in \sigma(r)} w_q^T x$ be the child that $x$ follows. Given $w, P, x, y$, the classification

loss at the current node $r$ is then

$$L(w, x, y, P) = 1 - P(\hat{q}, y).\tag{1}$$

Note that the final prediction of the example is made at a leaf node further down the tree, if the child to follow is not already a leaf node. Therefore $L$ is a lower bound of the actual loss. It is thus important to achieve a smaller $L$ because it could be a bottleneck of the final accuracy.

We measure efficiency by how fast the set of possible class labels shrinks. Efficiency is maximized when each child has a minimal number of class labels so that an unambiguous prediction can be made, otherwise we incur further cost for traveling down the tree. Given a test example, we define *ambiguity* as our efficiency measure, *i.e.* the size of label set of the child that the example follows, relative to its parent's size. Specifically, given $w$ and $P$, the ambiguity for an example $x$ is

$$A(w, x, P) = \frac{1}{K} \sum_{k=1}^{K} P(\hat{q}, k).\tag{2}$$

Note that $A \in [0, 1]$. A perfectly balanced $K$-nary tree would result in an ambiguity of $1/K$ for all examples at each node.

One important note is that the classification loss(accuracy) and ambiguity(efficiency) measures as defined in Eqn. 1 and Eqn. 2 are *local* to the current node being considered in greedily building the tree. They serve as proxies to the *global* accuracy and efficiency of the entire tree. For the rest of this paper, we will omit the "local" and "global" qualifications if it is clear according to the context.

Let $\epsilon > 0$ be the maximum classification loss we are willing to tolerate. Given a training set $(x_i, y_i), i = 1, \ldots, m$, we seek to minimize the average ambiguity of all examples while keeping the classification loss below $\epsilon$, which leads to the following optimization problem:

**OP1.** Optimizing efficiency with accuracy constraints.

$$\begin{aligned}
\underset{w,P}{\text{minimize}} \quad & \frac{1}{m} \sum_{i=1}^{m} A(w, x_i, P) \\
\text{subject to} \quad & \frac{1}{m} \sum_{i=1}^{m} L(w, x_i, y_i, P) \leq \epsilon \\
& P \in \{0, 1\}^{Q \times K}.
\end{aligned}$$

There are no further constraints on $P$ other than that its entries are integers 0 and 1. We do not require that the children cover all the classes in the parent. It is legal that one class in the parent can be assigned to none of the children, in which case we give up on the training examples from the class. In doing so, we pay a price on accuracy, *i.e.* those examples will have a misclassification loss of 1. Therefore a partition $P$ with all zeros is unlikely to be a good solution. We also allow overlap of label sets between children. If we cannot classify the examples from a class perfectly into one of the children, we allow them to go to more than one child. We pay a price on efficiency since we make less progress in eliminating possible class labels. This is different from the disjoint label sets in [1]. Overlapping label sets gives more flexibility and in fact leads to simpler optimization, as will become clear in Sec. 4.3.

Directly solving OP1 is intractable. However, with proper relaxation, we can alternate between optimizing over $w$ and over $P$ where each is a convex program.

## 4.2 Learning classifier weights $w$ given partitions $P$

Observe that fixing $P$ and optimizing over $w$ is similar to learning a multi-class classifier except for the overlapping classes. We relax the loss $L$ by a convex loss $\tilde{L}$ similar to the hinge loss.

$$\tilde{L}(w, x_i, y_i, P) = \max\{0, 1 + \max_{q \in A_i, r \in B_i} \{w_r^T x_i - w_q^T x_i)\}\}$$

where $A_i = \{q | P_{q, y_i} = 1\}$ and $B_i = \{r | P_{r, y_i} = 0\}$. Here $A_i$ is the set of children that contain class $y_i$ and $B_i$ is the rest of the children. The responses of the classifiers in $A_i$ are encouraged to be bigger than those in $B_i$, otherwise the loss $\tilde{L}$ increases. It is easily verifiable that $\tilde{L}$ upperbounds $L$. We then obtain the following convex optimization problem.

**OP2.** Optimizing over $w$ given $P$.

$$\underset{w}{\text{minimize}} \quad \lambda \sum_{q=1}^{Q} \|w_q\|_2^2 + \frac{1}{m} \sum_{i=1}^{m} \tilde{L}(w, x_i, y_i, P)$$

Note that here the objective is no longer the ambiguity $A$. This is because the influence of $w$ on $A$ is typically very small. When the partition $P$ is fixed, $w$ can lower $A$ by classifying examples into the child with the smallest label set. However, the way $w$ classifies examples is mostly constrained by the accuracy cap $\epsilon$, especially for small $\epsilon$. Empirically we also found that in optimizing $\tilde{L}$ over $w$, $A$ remains almost constant. Therefore for simplicity we assume that $A$ is constant w.r.t $w$ and the optimization becomes minimizing classification loss to move $w$ to the feasible region. We also added a regularization term $\sum_{q=1}^{Q} \|w_q\|_2^2$.

## 4.3 Determining partitions $P$ given classifier weights $w$

If we fix $w$ and optimize over $P$, rearranging terms gives the following integer program.

**OP3.** Optimizing over $P$.

$$\underset{P}{\text{minimize}} \quad A(P) = \sum_{q,k} P_{qk} \frac{1}{mK} \sum_{i=1}^{m} \mathbf{1}(\hat{q}_i = q)$$

$$\text{subject to} \quad 1 - \sum_{q,k} P_{qk} \frac{1}{m} \sum_{i=1}^{m} \mathbf{1}(\hat{q}_i = q \wedge y_i = k) \leq \epsilon$$

$$P_{qk} \in \{0, 1\}, \forall q, k.$$

Integer programming in general is NP-hard. However, for this integer program, we can solve it by relaxing it to a linear program and then taking the ceiling of the solution. We show that this solution is in fact near optimal by showing that the number of non-integers can be very few, due to the fact that the LP has few constraints other than that the variables lie in $[0, 1]$ and most of the $[0, 1]$ constraints will be active. Specifically we use Lemma 4.1(proof in supplementary materials) to bound the rounded LP solution in Theorem 4.2.

**Lemma 4.1.** *For LP problem*

$$\underset{x}{\text{minimize}} \quad c^T x$$

$$\text{subject to} \quad Ax \leq b$$

$$0 \leq x \leq 1,$$

*where $A \in \mathbb{R}^{m \times n}, m < n$, if it is feasible, then there exists an optimal solution with at most $m$ non-integer entries and such a solution can be found in polynomial time.*

**Theorem 4.2.** *Let $A^*$ be an optimal value of OP3. A solution $P'$ can be computed within polynomial time such that $A(P') \leq A^* + \frac{1}{K}$.*

*Proof.* We relax OP3 to an LP by replacing the constraint $P_{qk} \in \{0, 1\}, \forall q, k$ with $P_{qk} \in [0, 1], \forall q, k$. Apply Lemma 4.1 and we obtain an optimal solution $P''$ of the LP with at most 1 non-integer. We take the ceiling of the fraction and obtain an integer solution $P'$ to OP3. The value of the LP, a lower bound of $A^*$, increases by at most $\frac{1}{K}$, since $\frac{1}{mK} \sum_{i=1}^{m} \mathbf{1}(\hat{q}_i = q) \leq \frac{1}{K}, \forall q.$ $\square$

Note that the ambiguity is a quantity in $[0, 1]$ and $K$ is the number of classes. Therefore for large numbers of classes the rounded solution is almost optimal.

## 4.4 Summary of algorithm

Now all ingredients are in place for an iterative algorithm to build the tree, except that we need to initialize the partition $P$ or the weights $w$. We find that a random initialization of $P$ works well in practice. Specifically, for each child, we randomly pick one class, without replacement, from the

label set of the parent. That is, for each row of $P$, randomly pick a column and set the column to $1$. This is analogous to picking the cluster seeds in the K-means algorithm.

We summarize the algorithm for building one level of tree nodes in Algorithm 2. The procedure is applied recursively from the root. Note that each training example only affects classifiers on one path of the tree, hence the training cost is $O(mD \log K)$ for a balanced tree.

---

**Algorithm 2** Grow a single node $r$

---

**Input:** $Q, \epsilon$ and training examples classified into node $r$ by its ancestors.
Initialize $P$. For each child, randomly pick one class label from the parent, without replacement.
**for** $t = 1 \rightarrow T$ **do**
    Fix $P$, solve OP2 and update $w$.
    Fix $w$, solve OP3 and update $P$.
**end for**

---

### 4.5 Efficiency constrained formulations

As mentioned earlier, we can also optimize accuracy given explicit efficiency constraints. Let $\delta$ be the maximum ambiguity we can tolerate. Let OP1', OP2', OP3' be the counterparts of OP1, OP2 and OP3. We obtain OP1' by replacing $\epsilon$ with $\delta$ and switching $L(w, x_i, y_i, P)$ and $A(w, x_i, p)$ in OP1. OP2' is the same as OP2 because we also treat $A$ as constant and minimize the classification loss $L$ unconstrained. OP3' can also be formulated in a straightforward manner, and solved nearly optimally by rounding from LP(Theorem 4.3).

**Theorem 4.3.** *Let $L^*$ be the optimal value of OP3'. A solution $P'$ can be computed within polynomial time such that $L(P') \leq L^* + \max_k \psi_k$, where $\psi_k = \frac{1}{m} \sum_{i=1}^{m} \mathbf{1}(y_i = k)$, is the percentage of training examples from class $k$.*

*Proof.* We relax OP3' to an LP. Apply Lemma 4.1 and obtain an optimal solution $P''$ with at most 1 non-integer. We take the floor of $P''$ and obtain a feasible solution $P'$ to $OP3'$. The value of the LP, a lower bound of $L^*$, increases by at most $\max_k \psi_k$, since $\frac{1}{m} \sum_i \mathbf{1}(\hat{q}_i = q \wedge y_i = k) \leq \frac{1}{m} \sum_{i=1}^{m} \mathbf{1}(y_i = k) \leq \max_k \psi_k, \forall k, q.$     □

For uniform distribution of examples among classes, $\max_k \psi_k = 1/K$ and the rounded solution is near optimal for large $K$. If the distribution is highly skewed, for example, a heavy tail, then the rounding can give poor approximation. One simple workaround is to split the big classes into artificial subclasses or treat the classes in the tail as one big class, to "equalize" the distribution. Then the same learning techniques can be applied. In this paper we focus on the near uniform case and leave further discussion on the skewed case as future work.

## 5 Experiments

We use two datasets for evaluation: ILSVRC2010 [12] and ImageNet10K [6]. In ILSVRC2010, there are 1.2M images from 1k classes for training, 50k images for validation and 150k images for test. For each image in ILSVRC2010 we compute the LLC [13] feature with SIFT on a 10k codebook and use a two level spatial pyramid(1x1 and 2x2 grids) to obtain a 50k dimensional feature vector. In ImageNet10K, there are 9M images from 10184 classes. We use $50\%$ for training, $25\%$ for validation, and the rest $25\%$ for testing. For ImageNet10K, We compute LLC similarly except that we use no spatial pyramid, obtaining a 10k dimensional feature vector.

We use parallel stochastic gradient descent(SGD) [17] for training. SGD is especially suited for large scale learning [4] where the learning is bounded by the time and the features can no longer fit into memory (the LLC features take 80G in sparse format). Parallelization makes it possible to use multiple CPUs to improves wall time.

We compare our algorithm with the original label tree learning method by Bengio *et al.* [1]. For both algorithms, we fix two parameters, the number of children $Q$ for each node, and the maximum depth $H$ of the tree. The depth of each node is defined as the maximum distance to the root(the root

| | | $T_{32,2}$ | | | $T_{10,3}$ | | | $T_{6,4}$ | | | $T_{101,2}$ | | |
|---|---|---|---|---|---|---|---|---|---|---|---|---|---|
| | | $Acc\%$ | $C_{tr}$ | $S_{te}$ | $Acc\%$ | $C_{tr}$ | $S_{te}$ | $Acc\%$ | $C_{tr}$ | $S_{te}$ | $Acc\%$ | $C_{tr}$ | $S_{te}$ |
| Ours | | **11.9** | **259** | **10.3** | **8.92** | **104** | **18.2** | 5.62 | **50.2** | **31.3** | **3.4** | **685** | **32.4** |
| [1] | | 8.33 | 321 | **10.3** | 5.99 | 193 | 15.2 | **5.88** | 250 | 9.32 | 2.7 | 1191 | **32.4** |

Table 1: Global accuracy(Acc), training cost($C_{tr}$), and test speedup($S_{te}$) on ILSVRC2010 1K classes ($T_{32,2}, T_{10,3}, T_{6,4}$) and on ImageNet10K($T_{101,2}$) classes. Training and test costs are measured as the average number of vector operations performed per example. Test speedup is the one-vs-all test cost divided by the label tree test cost. Ours outperforms the Bengio *et al.* [1] approach by achieving comparable or better accuracy and efficiency, with less training cost, compared with the training cost for Bengio *et al.* [1] with the one-vs-all training cost excluded.

| Tree | | $T_{32,2}$ | | $T_{10,3}$ | | | $T_{6,4}$ | | | |
|---|---|---|---|---|---|---|---|---|---|---|
| Depth | | 0 | 1 | 0 | 1 | 2 | 0 | 1 | 2 | 3 |
| Classification loss(%) | Ours | 49.9 | 76.1 | 34.6 | 52.6 | 71.2 | 30.0 | 48.8 | 55.9 | 64.4 |
| | Bengio [1] | 76.6 | 64.8 | 62.8 | 53.7 | 65.3 | 56.2 | 34.8 | 37.3 | 65.8 |
| Ambiguity(%) | Ours | 6.49 | 1.55 | 18.9 | 18.4 | 2.96 | 24.7 | 24.1 | 23.5 | 7.15 |
| | Bengio [1] | 6.49 | 1.87 | 19.0 | 25.9 | 2.95 | 24.7 | 59.6 | 56.5 | 2.02 |

Table 2: Local classification loss(Eqn. 1) and ambiguity(Eqn. 2) measured at different depth levels for all trees on the ILSVRC2010 test set(1k classes). $T_{6,4}$ of Bengio *et al.* is less balanced(large ambiguity). Our trees are more balanced as efficiency is explicitly enforced by capping the ambiguity throughout all levels.

has depth 0). We require every internal node to split into $Q$ children, with two exceptions: nodes at depth $H-1$(parent of leaves) and nodes with fewer than $Q$ classes. In both cases, we split the node fully, *i.e.* grow one child node per class. We use $T_{Q,H}$ to denote a tree built with parameters $Q$ and $H$. We set $Q$ and $H$ such that for a well balanced tree, the number of leaf nodes $Q^H$ approximate the number of classes $K$.

We evaluate the *global* classification accuracy and computational cost in both training and test. The main costs of learning consist of two operations, evaluating the gradient and updating the weights, *i.e.* vector dot products and vector additions(possibly with scaling). We treat both operations as costing the same [1]. To measure the cost, we count the number of *vector* operations performed per training example. For instance, running SGD one-versus-all(either independent or single machine SVMs [5]) for $K$ classes costs $2K$ per example for going through data once, as in each iteration all $K$ classifiers are evaluated against the feature vector(dot product) and updated(addition).

For both algorithms, we build three trees $T_{32,2}, T_{10,3}, T_{6,4}$ for the ILSVRC2010 1k classes and build one tree $T_{101,2}$ for ImageNet10K classes. For the Bengio *et al.* method, we first train one-versus-all classifiers with one pass of parallel SGD. This results in a cost of 2000 per example for ISVRC2010 and 20368 for ImageNet10K. After forming the tree skeleton by spectral clustering using confusion matrix from the validation set, we learn the weights by solving a joint optimization(see [1]) with two passes of parallel SGD. For our method, we do three iterations in Algorithm 2. In each iteration, we do one pass of parallel SGD to solve OP3', such that the computation is comparable to that of Bengio *et al.* (excluding the one-versus-all training). We then solve OP3' on the validation set to update the partition. To set the efficiency constraint, we measure the average (local) ambiguity of the root node of the tree generated by the Bengio *et al.* approach, on the validation set. We use it as our ambiguity cap throughout our learning, in an attempt to produce a similarly structured tree.

We report the test results in Table 1. The results show that for all types of trees, our method achieves comparable or significantly better accuracy while achieving better speed-up with much less training cost, even after excluding the 1-versus-all training in Bengio *et al.*'s. It's worth noting that for the Bengio *et al.* approach, $T_{6,4}$ fails to further speed-up testing compared to the other shallower trees. The reason is that at depth 1(one level down from root), the splits became highly imbalanced and does not shrink the class sets faster enough until the height limit is reached. This is revealed in Table 2, where we measure the average local ambiguity(Eq. 2) and classification loss(Eq. 1) at each depth on the test set to shed more light on the structure of the trees. Observe that our trees have

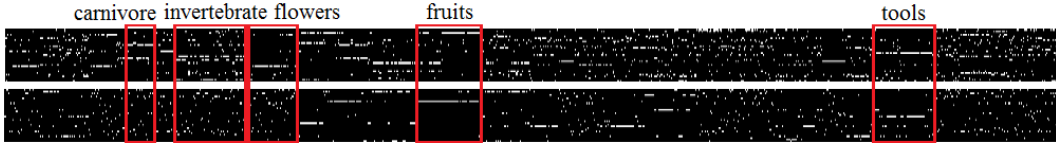

Figure 1: Comparison of partition matrices($32 \times 1000$) of the root node of $T_{32,2}$ for our approach(top) and the Bengio *et al.* approach(bottom). Each entry represents the membership of a class label(column) in a child(row). The columns are ordered by a depth first search of WordNet. Columns belonging to certain WordNet subtrees are marked by red boxes.

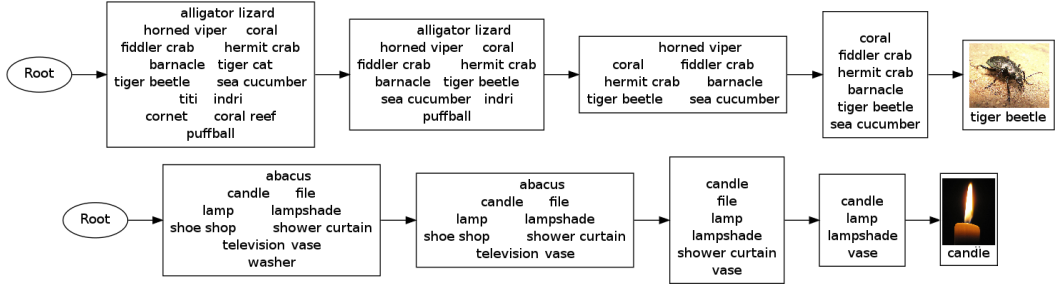

Figure 2: Paths of the tree $T_{6,4}$ taken by two test examples. The class labels shown are randomly subsampled to fit into the space.

almost constant average ambiguity at each level, as enforced in learning. This shows an advantage of our algorithm since we are able to explicitly enforce balanced tree while in Bengio *et al.* [1] no such control is possible, although spectral clustering encourages balanced splits.

In Fig. 1, we visualize the partition matrices of the root of $T_{32,2}$, for both algorithms. The columns are ordered by a depth first search of the WordNet tree so that neighboring columns are likely to be semantic similar classes. We observe that for both methods, there is visible alignment of the WordNet ordering. We further illustrate the semantic alignment by showing with the paths of our $T_{6,4}$ traveled by two test examples. Also observe that our partition is notably "noisier", despite that both partitions have the same average ambiguity. This is a result of overlapping partitions, which in fact improves accuracy(as shown in Table 2) because it avoids the mistakes made by forcing all examples of a class commit to one child.

Also note that Bengio *et al.* showed in [1] that optimizing the classifiers on the tree jointly is significantly better than independently training the classifiers for each node, as it encodes the dependency of the classifiers along a tree path. This does not contradict our results. Although we have no explicit joint learning of classifiers over the entire tree, we train the classifiers of each node using examples already filtered by classifiers of the ancestors, thus implicitly enforcing the dependency.

# 6 Conclusion

We have presented a novel approach to efficiently learn a label tree for large scale classification with many classes, allowing fine grained efficiency-accuracy tradeoff. Experimental results demonstrate more efficient trees at better accuracy with less training cost compared to previous work.

**Acknowledgment**

L. F-F is partially supported by an NSF CAREER grant (IIS-0845230), the DARPA CSSG grant, and a Google research award.

## Footnotes

[1]This is inconsequential as a vector addition always pairs with a dot product for all training in this paper.

# References

[1] S. Bengio, J. Weston, and D. Grangier. Label embedding trees for large multi-class tasks. In *Advances in Neural Information Processing Systems (NIPS)*, 2010.

[2] A. Beygelzimer, J. Langford, and P. Ravikumar. Multiclass classification with filter trees. *Preprint, June*, 2007.

[3] Alina Beygelzimer, John Langford, Yuri Lifshits, Gregory B. Sorkin, and Alexander L. Strehl. Conditional probability tree estimation analysis and algorithms. *Computing Research Repository*, 2009.

[4] L. Bottou and O. Bousquet. The tradeoffs of large scale learning. *Advances in neural information processing systems*, 20:161–168, 2008.

[5] K. Crammer and Y. Singer. On the algorithmic implementation of multiclass kernel-based vector machines. *The Journal of Machine Learning Research*, 2:265–292, 2002.

[6] J. Deng, A.C. Berg, K. Li, and L. Fei-Fei. What does classifying more than 10,000 image categories tell us? In *ECCV10*.

[7] J. Deng, W. Dong, R. Socher, L.J. Li, K. Li, and L. Fei-Fei. ImageNet: A large-scale hierarchical image database. In *CVPR09*, 2009.

[8] C. Fellbaum. *WordNet: An Electronic Lexical Database*. MIT Press, 1998.

[9] Gregory Griffin and Pietro Perona. Learning and using taxonomies for fast visual categorization. *CVPR08*, 2008.

[10] Y. Lin, F. Lv, S. Zhu, M. Yang, T. Cour, K. Yu, L. Cao, and T. Huang. Large-scale image classification: Fast feature extraction and svm training. In *Conference on Computer Vision and Pattern Recognition, page (to appear)*, volume 1, page 3, 2011.

[11] A. Torralba, R. Fergus, and W.T. Freeman. 80 million tiny images: A large data set for non-parametric object and scene recognition. *IEEE Transactions on Pattern Analysis and Machine Intelligence*, pages 1958–1970, 2008.

[12] http://www.image-net.org/challenges/LSVRC/2010/.

[13] J. Wang, J. Yang, K. Yu, F. Lv, T. Huang, and Y. Gong. Locality-constrained linear coding for image classification. 2010.

[14] D. Weiss, B. Sapp, and B. Taskar. Sidestepping intractable inference with structured ensemble cascades. In *NIPS*, volume 1281, pages 1282–1284, 2010.

[15] K. Yu and T. Zhang. Improved local coordinate coding using local tangents. *ICML09*, 2010.

[16] X. Zhou, K. Yu, T. Zhang, and T. Huang. Image classification using super-vector coding of local image descriptors. *Computer Vision–ECCV 2010*, pages 141–154, 2010.

[17] M. Zinkevich, M. Weimer, A. Smola, and L. Li. Parallelized stochastic gradient descent. In J. Lafferty, C. K. I. Williams, J. Shawe-Taylor, R.S. Zemel, and A. Culotta, editors, *Advances in Neural Information Processing Systems 23*, pages 2595–2603. 2010.

